# Probabilistic Visualisation of High-dimensional Binary Data

**Michael E. Tipping**

Microsoft Research,
St George House, 1 Guildhall Street,
Cambridge CB2 3NH, U.K.

mtipping@microsoft.com

## Abstract

We present a probabilistic latent-variable framework for data visualisation, a key feature of which is its applicability to binary and categorical data types for which few established methods exist. A variational approximation to the likelihood is exploited to derive a fast algorithm for determining the model parameters. Illustrations of application to real and synthetic binary data sets are given.

## 1  Introduction

Visualisation is a powerful tool in the exploratory analysis of multivariate data. The rendering of high-dimensional data in two dimensions, while generally implying loss of information, often reveals interesting structure to the human eye. Standard dimensionality-reduction methods from multivariate analysis, notably the principal component projection, are often utilised for this purpose, while techniques such as 'projection pursuit' have been tailored specifically to this end. With the current trend for larger databases and the need for effective 'data mining' methods, visualisation is becoming increasingly topical, and recent novel developments include nonlinear topographic methods (Lowe and Tipping 1997; Bishop, Svensén, and Williams 1998) and hierarchical combinations of linear models (Bishop and Tipping 1998). However, a disadvantageous aspect of many proposed techniques is their applicability only to continuous variables; there are very few such methods proposed specifically for the visualisation of discrete *binary* data types, which are commonplace in real-world datasets.

We approach this difficulty by proposing a probabilistic framework for the visualisation of arbitrary data types, based on an underlying latent variable density model. This leads to an algorithm which permits the visualisation of structure within data, while also defining a generative observation probability model. A further, and

intuitively pleasing, result is that the specialisation of the model to continuous variables recovers principal component analysis. Continuous, binary and categorical data types may thus be combined and visualised together within this framework, but for reasons of space, we concentrate on binary types alone in this paper.

In the next section we outline the proposed latent variable approach, and in Section 3 consider the difficulties involved in estimating the parameters in this model, giving an efficient variational scheme to this end in Section 4. In Section 5 we illustrate the application of the model and consider the accuracy of the variational approximation.

## 2 Latent Variable Models for Visualisation

In an ideal visualisation model, we would wish all of the dependencies between variables to be evident in the visualisation space, while the information that we lose in the dimensionality-reduction process should represent "noise", independent to each variable. This principle is captured by the following probability density model for a dataset comprising $d$-dimensional observation vectors $\mathbf{t} = (t_1, t_2, \ldots, t_d)$:

$$p(\mathbf{t}) = \int \left\{ \prod_{i=1}^{d} p(t_i|\mathbf{x}, \boldsymbol{\theta}) \right\} p(\mathbf{x})d\mathbf{x}, \tag{1}$$

where $\mathbf{x}$ is a two-dimensional *latent* variable vector, the distribution of which must be *a priori* specified, and $\boldsymbol{\theta}$ are the model parameters. Now, for a given value of $\mathbf{x}$ (or location in the visualisation space), the observations are independent under the model. (In general, of course, the model and conditional independence assumption will only hold approximately.) However, the unconditional observation model $p(\mathbf{t})$ does not, in general, factorise and so can still capture dependencies between the $d$ variables, given the constraint implied by the use of just two underlying latent variables. So, having estimated the parameters $\boldsymbol{\theta}$, data could be visualised by 'inverting' the generative model using Bayes' rule: $p(\mathbf{x}|\mathbf{t}) = p(\mathbf{t}|\mathbf{x})p(\mathbf{x})/p(\mathbf{t})$. Each data point then induces a distribution in the latent space, which for the purposes of visualisation, we might summarise with the conditional mean value $\langle \mathbf{x}|\mathbf{t} \rangle$.

That this form of model can be appropriate for visualisation was demonstrated by Bishop and Tipping (1998), who showed that if the latent variables are defined to be independent and Gaussian, $\mathbf{x} \sim \mathcal{N}(\mathbf{0}, \mathbf{I})$, and the conditional observation model is also Gaussian, $t_i|\mathbf{x} \sim \mathcal{N}(\mathbf{w}_i^T\mathbf{x} + \mu_i, \sigma_i^2\mathbf{I})$, then maximum-likelihood estimation of the model parameters $\{\mathbf{w}_i, \mu_i, \sigma_i^2\}$ leads to a model where the the posterior mean $\langle \mathbf{x}|\mathbf{t} \rangle$ is equivalent to a probabilistic principal component projection.

A visualisation method for binary variables now follows naturally. Retaining the Gaussian latent distribution $\mathbf{x} \sim \mathcal{N}(\mathbf{0}, \mathbf{I})$, we specify an appropriate conditional distribution for $P(t_i|\mathbf{x}, \boldsymbol{\theta})$. Given that principal components corresponds to a linear model for continuous data types, we adopt the appropriate generalised linear model in the binary case:

$$P(t_i|\mathbf{x}) = \sigma(A_i)^{t_i} \left\{ 1 - \sigma(A_i) \right\}^{1-t_i}, \tag{2}$$

where $\sigma(A) = \{1 + \exp(-A)\}^{-1}$ and $A_i = \mathbf{w}_i^T\mathbf{x} + b_i$ with parameters $\mathbf{w}_i$ and $b_i$.

## 3 Maximum-likelihood Parameter Estimation

The proposed model for binary data already exists in the literature under various guises, most historically as a *latent trait* model (Bartholomew 1987), although it is not utilised for data visualisation. While in the case of probabilistic principal

component analysis, ML parameter estimates can be obtained in closed-form, a disadvantageous feature of the binary model is that, with $P(t_i|\mathbf{x})$ defined by (2), the integral of (1) is analytically intractable and $P(\mathbf{t})$ cannot be computed directly. Fitting a latent trait model thus necessitates a numerical integration, and recent papers have considered both Gauss-Hermite (Moustaki 1996) and Monte-Carlo sampling approximations (Mackay 1995; Sammel, Ryan, and Legler 1997).

In this latter case, the log-likelihood for a dataset of $N$ observation vectors $\{\mathbf{t}_1, \ldots, \mathbf{t}_N\}$ would be approximated by

$$\mathcal{L} \approx \sum_{n=1}^{N} \ln \left\{ \frac{1}{L} \sum_{l=1}^{L} \prod_{i=1}^{d} P(t_{in}|\mathbf{x}_l, \mathbf{w}_i, b_i) \right\} \tag{3}$$

where $\mathbf{x}_l$, $l = 1 \ldots L$, are samples from the two-dimensional latent distribution.

To obtain parameter estimates we may utilise an expectation-maximisation (EM) approach by noting that (3) is equivalent in form to an $L$-component *latent class* model (Bartholomew 1987) where the component probabilities are mutually constrained from (2). Applying standard methodology leads to an E-step which requires computation of $N \times L$ posterior 'responsibilities' $P(\mathbf{x}_l|\mathbf{t}_n)$, and a logistic regression M-step which is unfortunately iterative, although it can be performed relatively efficiently by an iteratively re-weighted least-squares algorithm. Because of these difficulties in implementation, in the next section we describe a variational approximation to the likelihood which can be maximised more efficiently.

## 4   A Variational Approximation to the Likelihood

Jaakkola and Jordan (1997) introduced a variational approximation for the predictive likelihood in a Bayesian logistic regression model and also briefly considered the "dual" problem, which is closely related to the proposed visualisation model. In this approach, the integral in (1) is approximated by:

$$\widetilde{P}(\mathbf{t}) = \int \left\{ \prod_{i=1}^{d} \widetilde{P}(t_i|\mathbf{x}, \xi_i) \right\} p(\mathbf{x}) \, d\mathbf{x}, \tag{4}$$

where

$$\widetilde{P}(t_i|\mathbf{x}, \xi_i) = \sigma(\xi_i) \exp \left\{ (A_i - \xi_i)/2 + \lambda(\xi_i)(A_i^2 - \xi_i^2) \right\}, \tag{5}$$

with $A_i = (2t_i - 1)(\mathbf{w}_i^{\mathsf{T}}\mathbf{x} + b_i)$ and $\lambda(\xi_i) = \{0.5 - \sigma(\xi_i)\}/2\xi_i$. The parameters $\xi_i$ are the 'variational' parameters, and this approximation has the property that $\widetilde{P}(t_i|\mathbf{x}, \xi_i) \leq P(t_i|\mathbf{x})$, with equality at $\xi_i = A_i$, and thus it follows that $\widetilde{P}(\mathbf{t}) \leq P(\mathbf{t})$.

Now because the exponential in (5) is quadratic in $\mathbf{x}$, then the integral in (4), and also the likelihood, can be computed in closed form. This suggests an alternative algorithm for finding parameter estimates where we iteratively maximise the variational approximation to the likelihood. Each iteration of this algorithm is guaranteed to increase a lower bound on, but will not necessarily maximise, the true likelihood. Nevertheless, we would hope that it will be a close approximation, the accuracy of which is investigated later. At each step in the algorithm, then, we:

1. Obtain the sufficient statistics for the approximated posterior distribution of latent variables given each observation, $\widetilde{p}(\mathbf{x}_n|\mathbf{t}_n, \boldsymbol{\xi}_n)$.

2. Optimise the variational parameters $\xi_{in}$ in order to make the approximation $\widetilde{P}(\mathbf{t}_n)$ as close as possible to $P(\mathbf{t}_n)$ for all $\mathbf{t}_n$.

3. Update the model parameters $\mathbf{w}_i$ and $b_i$ to increase $\widetilde{P}(\mathbf{t})$.

Jaakkola and Jordan (1997) give formulae for the above computations, but these do not include provision for the 'biases' $b_i$, and so the necessary expressions are re-derived below. Note that although we have introduced $N \times d$ additional variational parameters, it is no longer necessary to sample from $p(\mathbf{x})$ and compute responsibilities, and no iterative logistic regression step is needed.

**Computing the Latent Posterior Statistics.** From Bayes' rule, the posterior approximation $\widetilde{p}(\mathbf{x}_n|\mathbf{t}_n, \boldsymbol{\xi}_n)$ is Gaussian with covariance and mean given by

$$\mathbf{C}_n = \left[\mathbf{I} - 2\sum_{i=1}^{d} \lambda(\xi_{in})\mathbf{w}_i\mathbf{w}_i^\mathsf{T}\right]^{-1}, \tag{6}$$

$$\boldsymbol{\mu}_n = \mathbf{C}_n \left\{\sum_{i=1}^{d}\left[t_{in} - \frac{1}{2} + 2\lambda(\xi_{in})b_i\right]\mathbf{w}_i\right\}. \tag{7}$$

**Optimising the Variational Parameters.** Because $P(\mathbf{t}) \geq \widetilde{P}(\mathbf{t})$, the variational approximation can be optimised by maximising $\widetilde{P}(\mathbf{t}_n)$ with respect to each $\xi_{in}$. We use the EM methodology to obtain updates

$$\xi_{in}^2 = \mathbf{w}_i^\mathsf{T}\langle\mathbf{x}_n\mathbf{x}_n^\mathsf{T}\rangle\mathbf{w}_i + 2b_i\mathbf{w}_i^\mathsf{T}\langle\mathbf{x}_n\rangle + b_i^2, \tag{8}$$

where the angle brackets $\langle\cdot\rangle$ denote expectations with respect to $\widetilde{p}(\mathbf{x}_n|\mathbf{t}_n, \boldsymbol{\xi}_n^\mathrm{old})$ and where, from (6) and (7) earlier, the necessary posterior statistics are given by:

$$\langle\mathbf{x}_n\rangle = \boldsymbol{\mu}_n, \tag{9}$$

$$\langle\mathbf{x}_n\mathbf{x}_n^\mathsf{T}\rangle = \mathbf{C}_n + \boldsymbol{\mu}_n\boldsymbol{\mu}_n^\mathsf{T}. \tag{10}$$

Since (6) and (7) depend on the variational parameters, $\mathbf{C}_n$ and $\boldsymbol{\mu}_n$ are computed followed by the update for each $\xi_{in}$ from (8). Iteration of this two-stage process is guaranteed to improve monotonically the approximation of $P(\mathbf{t}_n)$ and typically only two iterations are necessary for convergence.

**Optimising the Model Parameters.** We again use EM to increase the variational likelihood approximation with respect to $\mathbf{w}_i$ and $b_i$. Defining

$$\widehat{\mathbf{w}}_i = (\mathbf{w}_i^\mathsf{T}, b_i)^\mathsf{T},$$

$$\widehat{\mathbf{x}} = (\mathbf{x}^\mathsf{T}, 1)^\mathsf{T},$$

leads to updates for both $\mathbf{w}_i$ and $b_i$ given by:

$$\widehat{\mathbf{w}}_i = -\left[\sum_{n=1}^{N} 2\lambda(\xi_{in})\langle\widehat{\mathbf{x}}_n\widehat{\mathbf{x}}_n^\mathsf{T}\rangle\right]^{-1}\left[\sum_{n=1}^{N}(t_{in} - 1/2)\langle\widehat{\mathbf{x}}_n\rangle\right], \tag{11}$$

where

$$\langle\widehat{\mathbf{x}}_n\widehat{\mathbf{x}}_n^\mathsf{T}\rangle = \begin{pmatrix} \mathbf{C}_n + \boldsymbol{\mu}_n\boldsymbol{\mu}_n^\mathsf{T} & \boldsymbol{\mu}_n \\ \boldsymbol{\mu}_n^\mathsf{T} & 1 \end{pmatrix}. \tag{12}$$

## 5 Visualisation

**Synthetic clustered data.** We firstly give an example of visualisation of artificially-generated data to illustrate the operation and features of the method. Binary data was synthesised by first generating three random 16-bit prototype vectors, where each bit was set with probability 0.5. Next a 600-point dataset was generated by taking 200 examples of each prototype and inverting each bit with

probability 0.05. We generated a second dataset in the same manner, but where the probability of bit inversion was 0.15, simulating more "noise" about each prototype. The final values of $\mu_n$ from (7) for each data point are plotted in Figure 1. In the left plot for the low-noise dataset, the three clusters are clear, as are the prototype vectors. On the right, the bit-noise is sufficiently high such that clusters now overlap to a degree and the prototypes are no longer evident. However, we can elucidate further information from the model by drawing lines representing $P(t_i|\mathbf{x}) = 0.5$, or $\mathbf{w}_i^T\mathbf{x} + b_i = 0$, which may be considered to be 'decision boundaries' for each bit. These offer more convincing evidence of the presence of three clusters.

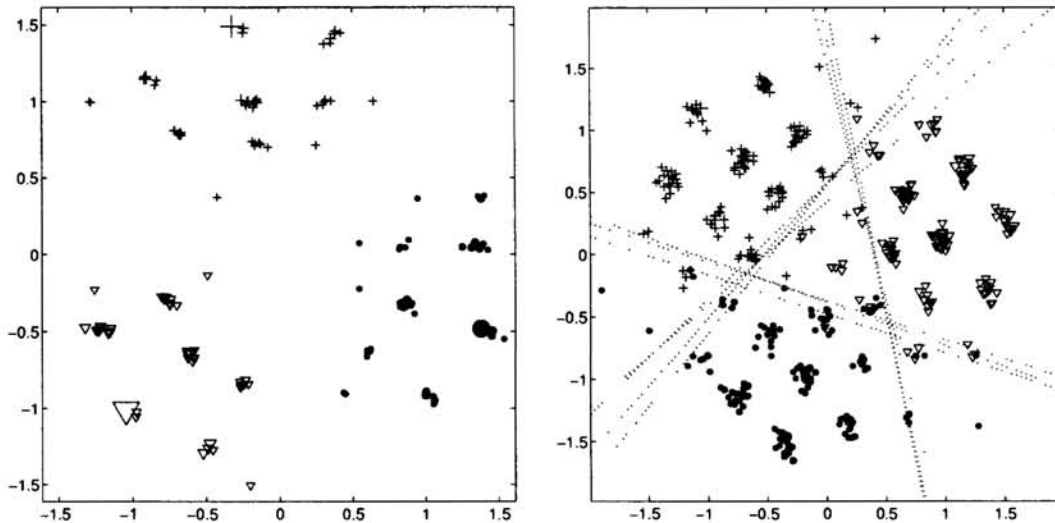

Figure 1: Visualisation of two synthetic clustered datasets. The three clusters have been denoted by separate glyphs, the size of which reflects the number of examples whose posterior means are located at that point in the latent space. In the right plot, lines corresponding to $P(t_i|\mathbf{x}) = 0.5$ have been drawn.

**Handwritten digit data.** On the left of Figure 2, a visualisation is given of 1000 examples derived from $16 \times 16$ images of handwritten digit '2's. There is visual evidence of the natural variability of writing styles in the plot as the posterior latent means in Figure 2 describe an approximate 'horseshoe' structure. On the right of the figure we examine the nature of this by plotting gray-scale images of the vectors $P(\mathbf{t}|\mathbf{x}_j)$, where $\mathbf{x}_j$ are four numbered samples in the visualisation space. These images illustrate the expected value of each bit given the latent-space location and demonstrate that the location is indeed indicative of the style of the digit, notably the presence of a loop.

**Accuracy of the variational approximation.** To investigate the accuracy of the approximation, the sampling algorithm of Section 3 for likelihood maximisation was implemented and applied to the above two datasets. The evolution of error (negative log-likelihood per data-point) was plotted against time for both algorithms, using identical initialisations. The 'true' error for the variational approach was estimated using the same 500-point Monte-Carlo sample. Typical results are shown in Figure 3, and the final running time and error (using a sensible stopping criterion) are given for both datasets in Table 1.

For these two example datasets, the variational algorithm converges considerably more quickly than in the sampling case, and the difference in final error is relatively small, particularly so for the larger-dimensionality dataset. The approximation of the posterior distributions $p(\mathbf{x}_n|\mathbf{t}_n)$ is the key factor in the accuracy of the algorithm. In Figure 4, contours of the posterior distribution in the latent space induced

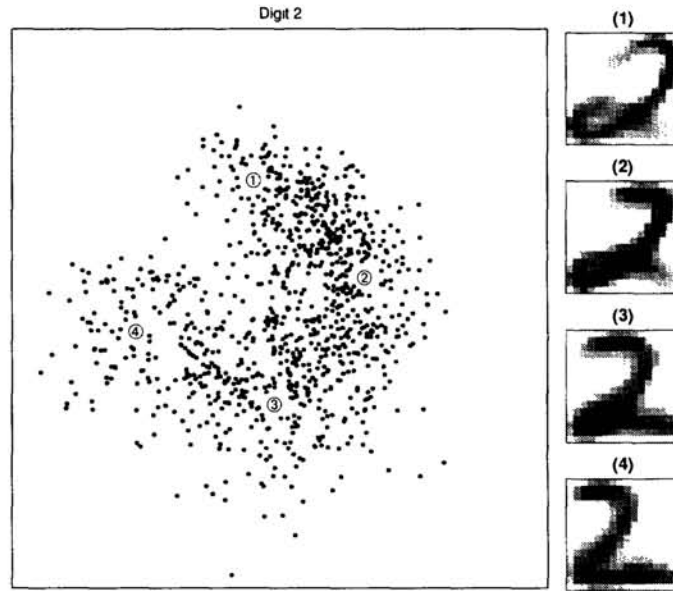

Figure 2: Left: visualisation of 256-dimensional digit '2' data. Right: gray-scale images of the conditional probability of each bit at the latent space locations marked.

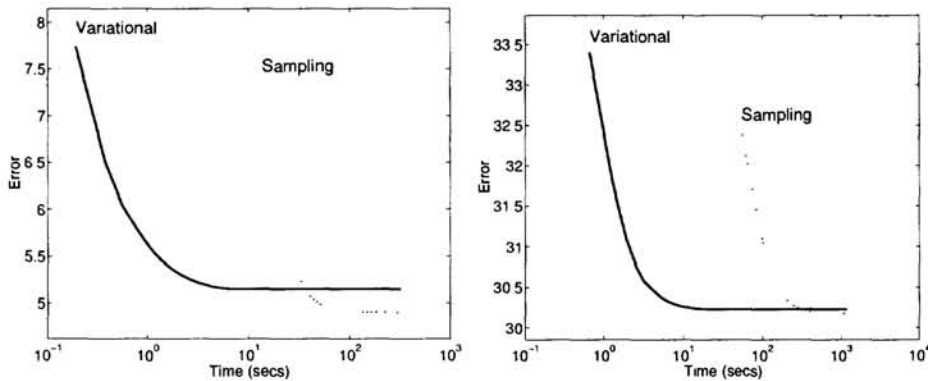

Figure 3: Error vs. time for the synthetic data (left) and the digit '2' data (right).

by a typical data point are shown for both algorithms and datasets. This approximation is more accurate as dimensionality increases (a phenomenon observed with other datasets too), as the true posterior becomes more Gaussian in form.

# 6 Conclusions

We have outlined a variational approximation for parameter estimation in a probabilistic visualisation model and although we have only considered its application to binary variables here, the extension to mixtures of arbitrary data types is readily implemented. For the two comparisons shown (and others not illustrated here), the approximation appears acceptably accurate, and particularly so for data of higher dimensionality. The algorithm is considerably faster than a sampling approach, which would permit incorporation of multiple models in a more complex hierarchical architecture, of a sort that has been effectively implemented for visualisation of continuous variables (Bishop and Tipping 1998).

|            | Synthetic-16 ||  Digit-256 ||
|            | Time  | Error | Time   | Error |
|------------|-------|-------|--------|-------|
| **Variational** | 7.8   | 5.14  | 25.6   | 30.23 |
| **Sampling**    | 331.1 | 4.93  | 1204.5 | 30.19 |

Table 1: Comparison of final error and running time for the two algorithms.

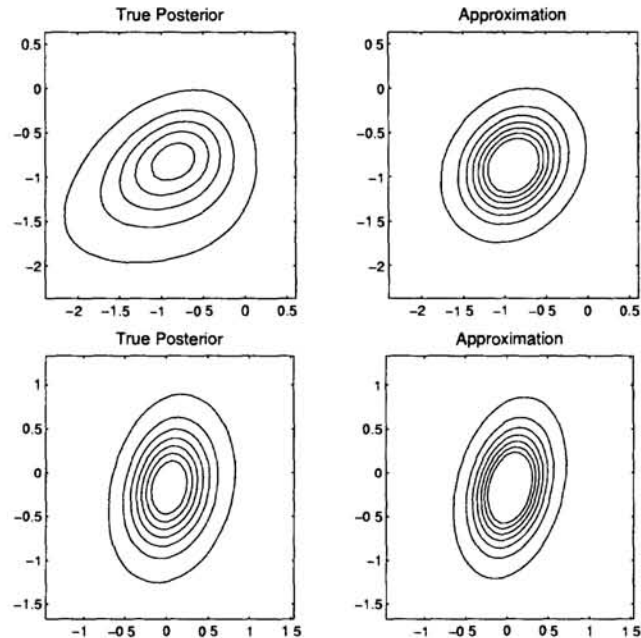

Figure 4: True and approximated posteriors for a single example from the synthetic data set (top) and the digit '2' data (bottom).

## 7   References

Bartholomew, D. J. (1987). *Latent Variable Models and Factor Analysis*. London: Charles Griffin & Co. Ltd.

Bishop, C. M., M. Svensén, and C. K. I. Williams (1998). GTM: the Generative Topographic Mapping. *Neural Computation 10*(1), 215–234.

Bishop, C. M. and M. E. Tipping (1998). A hierarchical latent variable model for data visualization. *IEEE Transactions on Pattern Analysis and Machine Intelligence 20*(3), 281–293.

Jaakkola, T. S. and M. I. Jordan (1997). Bayesian logistic regression: a variational approach. In D. Madigan and P. Smyth (Eds.), *Proceedings of the 1997 Conference on Artificial Intelligence and Statistics*, Ft Lauderdale, FL.

Lowe, D. and M. E. Tipping (1997). Neuroscale: Novel topographic feature extraction with radial basis function networks. In M. Mozer, M. Jordan, and T. Petsche (Eds.), *Advances in Neural Information Processing Systems 9*, pp. 543–549. Cambridge, Mass: MIT Press.

Mackay, D. J. C. (1995). Bayesian neural networks and density networks. *Nuclear Instruments and Methods in Physics Research, Section A 354*(1), 73–80.

Moustaki, I. (1996). A latent trait and a latent class model for mixed observed variables. *British Journal of Mathematical and Statistical Psychology 49*, 313–334.

Sammel, M. D., L. M. Ryan, and J. M. Legler (1997). Latent variable models for mixed discrete and continuous outcomes. *Journal of the Royal Statistical Society, Series B 59*, 667–678.
